# On Ranking in Survival Analysis: Bounds on the Concordance Index

**Vikas C. Raykar, Harald Steck, Balaji Krishnapuram**
CAD and Knowledge Solutions (IKM CKS), Siemens Medical Solutions Inc., Malvern, USA
{vikas.raykar,harald.steck,balaji.krishnapuram}@siemens.com

**Cary Dehing-Oberije, Philippe Lambin**
Maastro Clinic, University Hospital Maastricht, University Maastricht, GROW, The Netherlands
{cary.dehing,philippe.lambin}@maastro.nl

## Abstract

In this paper, we show that classical survival analysis involving censored data can naturally be cast as a ranking problem. The concordance index (CI), which quantifies the quality of rankings, is the standard performance measure for model *assessment* in survival analysis. In contrast, the standard approach to *learning* the popular proportional hazard (PH) model is based on Cox's partial likelihood. We devise two bounds on CI–one of which emerges directly from the properties of PH models–and optimize them *directly*. Our experimental results suggest that all three methods perform about equally well, with our new approach giving slightly better results. We also explain why a method designed to maximize the Cox's partial likelihood also ends up (approximately) maximizing the CI.

## 1  Introduction

Survival analysis is a well-established field in medical statistics concerned with analyzing/predicting the time until the occurrence of an event of interest–e.g., death, onset of a disease, or failure of a machine. It is applied not only in clinical research, but also in epidemiology, reliability engineering, marketing, insurance, etc. The time between a well-defined starting point and the occurrence of the event is called the *survival time* or *failure time*, measured in clock time or in another appropriate scale, e.g., mileage of a car. Survival time data are not amenable to standard statistical methods because of its two special features–(1) the continuous survival time often follows a skewed distribution, far from normal, and (2) a large portion of the data is censored (see Sec. 2). In this paper we take a machine learning perspective and cast survival analysis as a *ranking problem*–where the task is to rank the data points based on their survival times rather than to predict the actual survival times. One of the most popular performance measures for assessing learned models in survival analysis is the Concordance Index (CI), which is similar to the Wilcoxon-Mann-Whitney statistic [13, 10] used in bi-partite ranking problems.

Given the CI as a performance measure, we develop approaches that learn models by directly optimizing the CI. As optimization of the CI is computationally expensive, we focus on maximizing two lower bounds on the CI, namely the log-sigmoid and the exponential bounds, which are described in Sec. 4, 5, and 6. Interestingly, the log-sigmoid bound arises in a natural way from the Proportional Hazard (PH) model, which is the standard model used in classical survival analysis, see Sec. 5.2. Moreover, as the PH models are learned by optimizing Cox's partial likelihood in classical survival analysis, we show in Sec. 8 that maximizing this likelihood also ends up (approximately) maximizing the CI. Our experiments in Sec. 9 show that optimizing our two lower bounds and Cox's likelihood yields very similar results with respect to the CI, with the proposed lower bounds being slightly better.

## 2 Survival analysis

Survival analysis has been extensively studied in the statistics community for decades, e.g., [4, 8]. A primary focus is to build statistical models for survival time $T_i^*$ of individual $i$ of a population.

### 2.1 Censored data

A major problem is the fact that the *period of observation* $C_i^*$ can be censored for many individuals $i$. For instance, a patient may move to a different town and thus be no longer available for a clinical trial. Also at the end of the trial a lot of patients may actually survive. For such cases the *exact survival time may be longer than the observation period*. Such data are referred to as *right-censored*, and $C_i^*$ is also called the *censoring time*. For such individuals, we only know that they survived for *at least* $C_i^*$, *i.e.*, our actual observation is $T_i = \min(T_i^*, C_i^*)$.

Let $x_i \in \mathbf{R}^d$ be the associated $d$-dimensional vector of *covariates* (explanatory variables) for the $i^{\text{th}}$ individual. In clinical studies, the covariates typically include demographic variables, such as age, gender, or race; diagnosis information like lab tests; or treatment information, e.g., dosage. An important assumption generally made is that $T_i^*$ and $C_i^*$ are independent conditional on $x_i$, *i.e.*, the cause for censoring is independent of the survival time. With the indicator function $\delta_i$, which equals 1 if failure is observed ($T_i^* \leq C_i^*$) and 0 if data is censored ($T_i^* > C_i^*$), the available training data can be summarized as $\mathcal{D} = \{T_i, x_i, \delta_i\}_{i=1}^N$ for $N$ patients. The objective is to learn a predictive model for the survival time as a function of the covariates.

### 2.2 Failure time distributions

The failures times are typically modeled to follow a distribution, which absorbs both truly random effects and causes unexplained by the (available) covariates. This distribution is characterized by the *survival function* $S(t) = \Pr[T > t]$ for $t > 0$, which is the probability that the individual is still alive at time $t$. A related function commonly used is the *hazard function*. If $T$ has *density function* $p$, then the hazard function is defined by $\lambda(t) = \lim_{\Delta t \to 0} \Pr[t < T \leq t + \Delta t | T > t]/\Delta t = p(t)/S(t)$. The hazard function measures the instantaneous *rate* of failure, and provides more insight into the failure mechanisms. The function $\Lambda(t) = \int_0^t \lambda(u)du$ is called the *cumulative hazard function*, and it holds that $S(t) = e^{-\Lambda(t)}$ [4].

### 2.3 Proportional hazard model

Proportional hazard (PH) models have become the standard for studying the effect of the covariates on the survival time distributions, e.g., [8]. Specifically, the PH model assumes a multiplicative effect of the covariates on the hazard function, *i.e.*,

$$\lambda(t|x) = \lambda_0(t)e^{w^\top x}, \tag{1}$$

where $\lambda(t|x)$ is the hazard function of a person with covariates $x$; $\lambda_0(t)$ is the so-called baseline hazard function (*i.e.*, when $x = 0$), which is typically based on the exponential or the Weibull distributions; $w$ is a set of unknown regression parameters, and $e^{w^\top x}$ is the relative hazard function. Equivalent formulations for the cumulative hazard function and the survival function include

$$\Lambda(t|x) = \Lambda_0(t)e^{w^\top x}, \quad \text{and} \quad S(t|x) = e^{-\Lambda_0(t)e^{w^\top x}} = e^{-\left[e^{w^\top x} \int \lambda_0(t)dt\right]}. \tag{2}$$

### 2.4 Cox's partial likelihood

Cox noticed that a semi-parametric approach is sufficient for estimating the weights $w$ in PH models [2, 3], *i.e.*, the baseline hazard function can remain completely unspecified. Only a parametric assumption concerning the effect of the covariates on the hazard function is required. Parameter estimates in the PH model are obtained by maximizing *Cox's partial likelihood* (of the weights) [2, 3]:

$$L(w) = \prod_{T_i \text{ uncensored}} \frac{e^{w^\top x_i}}{\sum_{T_j \geq T_i} e^{w^\top x_j}}. \tag{3}$$

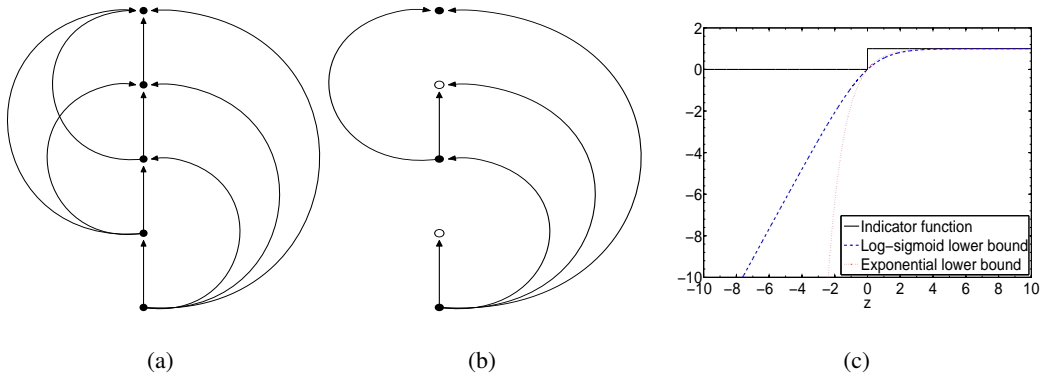

<div align="center">(a)         (b)         (c)</div>

Figure 1: Order graphs representing the ranking constraints. (a) No censored data and (b) with censored data. The empty circle represents a censored point. The points are arranged in the increasing value of their survival times with the lowest being at the bottom. (c) Two concave lower bounds on the 0-1 indicator function.

Each term in the product is the probability that the $i^{\text{th}}$ individual failed at time $T_i$ given that exactly one failure has occurred at time $T_i$ and all individuals for which $T_j \geq T_i$ are at risk of failing. Cox and others have shown that this partial log-likelihood can be treated as an ordinary log-likelihood to derive valid (partial) maximum likelihood estimates of $w$ [2, 3].

The interesting properties of the Cox's partial likelihood include: (1) due to its parametric form, it can be optimized in a computationally efficient way; (2) it depends only on the ranks of the observed survival times, cf. the inequality $T_j \geq T_i$ in Eq. 3, rather than on their actual numerical values. We outline this connection to the ranking of the times $T_i$–and hence the concordance index–in Sec. 8.

## 3 Ordering of Survival times

Casting survival analysis as ranking problem is an elegant way of dealing not only with the typically skewed distributions of survival times, but also with the censoring of the data: Two subjects' survival times can be ordered not only if (1) both of them are uncensored but also if (2) the uncensored time of one is smaller than the censored survival time of the other. This can be visualized by means of an order graph $\mathcal{G} = (\mathcal{V}, \mathcal{E})$, cf. also Fig. 1. The set of vertices $\mathcal{V}$ represents all the individuals, where each filled vertex indicates an *observed/uncensored* survival time, while an empty circle denotes a *censored* observation. Existence of an edge $\mathcal{E}_{ij}$ implies that $T_i < T_j$. An edge cannot originate from a censored point.

### 3.1 Concordance index

For these reasons, the *concordance index* (CI) or $c$-index is one of the most commonly used performance measures of survival models, e.g., [6]. It can be interpreted as the fraction of all pairs of subjects whose predicted survival times are correctly ordered among all subjects that can actually be ordered. In other words, it is the probability of concordance between the predicted and the observed survival. It can be written as

$$c(\mathcal{D}, \mathcal{G}, f) = \frac{1}{|\mathcal{E}|} \sum_{\mathcal{E}_{ij}} \mathbf{1}_{f(x_i) < f(x_j)} \tag{4}$$

with the indicator function $\mathbf{1}_{a<b} = 1$ if $a < b$, and 0 otherwise; $|\mathcal{E}|$ denotes the number of edges in the order graph. $f(x_i)$ is the predicted survival time for subject $i$ by the model $f$. Equivalently, the concordance index can also be written explicitly as

$$c = \frac{1}{|\mathcal{E}|} \sum_{T_i \text{ uncensored}} \sum_{T_j > T_i} \mathbf{1}_{f(x_i) < f(x_j)}. \tag{5}$$

This index is a generalization of the Wilcoxon-Mann-Whitney statistics [13, 10] and thus of the area under the ROC curve (AUC) to regression problems in that it can (1) be applied to continuous

<div align="center">3</div>

output variables and (2) account for censoring of the data. Like for the AUC, $c = 1$ indicates perfect prediction accuracy and $c = 0.5$ is as good as a random predictor.

## 3.2 Maximizing the CI—The Ranking Problem

Since we evaluate the predictive accuracy of a survival model in terms of the concordance index, it is natural to formulate the learning problem to directly maximize the concordance index. Note that, while the concordance index has been used widely to evaluate a learnt model, it is not generally used as an objective function during training. As the concordance index is invariant to any monotone transformation of the survival times, the model learnt by maximizing the $c$-index is actually a *ranking/scoring* function. Our goal is to predict whether the survival time of one individual is larger than the one of another individual. Very often the doctor would like to know whether a particular kind of treatment results in an increase in the survival time and the exact absolute value of the survival time is not important. In terms of ranking problems studied in machine learning this is an $N$-*partite ranking problem*, where every data point is a class in itself. Formulating it as a ranking problem allows us to naturally incorporate the censored data. Once we have formulated it as a ranking problem we can use various ranking algorithms proposed in the machine learning literature [5, 7, 1, 12]. In this paper we use the algorithm proposed by [12].

More formally, we would like to learn a *ranking* function $f$ from a suitable function class $\mathcal{F}$, such that $f(x_i) > f(x_j)$ implies that the survival time of patient $i$ is larger than the one of patient $j$. Given the data $\mathcal{D}$ and the order graph $\mathcal{G}$, the optimal ranking function is $\widehat{f} = \arg \max_{f \in \mathcal{F}} c(\mathcal{D}, \mathcal{G}, f)$. As to prevent overfitting on the training data, regularization can be added to this equation, see Secs. 5 and 6. In many cases, sufficient regularization is also achieved by restricting the function class $\mathcal{F}$, e.g., it may contain only linear functions. For ease of exposition we will consider the family of linear ranking functions [1] in this paper: $\mathcal{F} = \{f_w\}$, where for any $x, w \in \mathcal{R}^d$, $f_w(x) = w^\top x$.

# 4 Lower bounds on the CI

Maximizing the CI is a discrete optimization problem, which is computationally expensive. For this reason, we resort to maximizing a differentiable and concave lower bound on the 0-1 indicator function in the concordance index, cf. Eqs. 4 and 5. In this paper we focus on the *log-sigmoid lower bound* [12], cf. Sec. 5, and *exponential lower bound*, cf. Sec. 6, which are suitably scaled as to be tight at the origin and also in the asymptotic limit of large positive values, see also Fig. 1(c). We will also show how these bounds relate to the classical approaches in survival analysis: as it turns out, for the family of linear ranking functions, these two approaches are closely related to the PH model commonly used in survival analysis, cf. Sec. 5.2.

# 5 Log-sigmoid lower bound

The first subsection discusses the lower bound on the concordance index based on the log-sigmoid function. The second subsection shows that this bound arises naturally when using proportional hazard models.

## 5.1 Lower bound

The sigmoid function is defined as $\sigma(z) = 1/(1+e^{-z})$, While it is an *approximation* to the indicator function, it is not a *lower bound*. In contrast, the scaled version of the log of the sigmoid function, $\log\left[2\sigma(z)\right]/\log 2$, is a lower bound on the indicator function (Fig. 1(c)), *i.e.*,

$$\mathbf{1}_{z>0} \geq 1 + (\log \sigma(z)/\log 2). \tag{6}$$

The log-sigmoid function is concave and asymptotically linear for large negative values, and may hence be considered a differentiable approximation to the hinge loss, which is commonly used for

training support vector machines. The lower bound on the concordance index (cf. Eq. 4) follows immediately:

$$c = \frac{1}{|\mathcal{E}|} \sum_{\mathcal{E}_{ij}} \mathbf{1}_{f(x_j)-f(x_i)>0} \geq \frac{1}{|\mathcal{E}|} \sum_{\mathcal{E}_{ij}} 1 + (\log \sigma[f(x_j) - f(x_i)]/\log 2) \equiv \widehat{c}_{\text{LS}}, \qquad (7)$$

which can efficiently be maximized by gradient-based methods (cf. Sec 7). Given the linear ranking function $f_w(x) = w^\top x$, the bound $\widehat{c}_{\text{LS}}$ becomes

$$\widehat{c}_{\text{LS}}(w) = \frac{1}{|\mathcal{E}|} \sum_{\mathcal{E}_{ij}} 1 + (\log \sigma[w^\top(x_j - x_i)]/\log 2). \qquad (8)$$

As to avoid overfitting, we penalize functions with a large norm $w$ in the standard way, and obtain the regularized version

$$\widehat{c}_{\text{LSreg}}(w) = -\frac{\lambda}{2}\|w\|^2 + \widehat{c}_{\text{LS}}(w). \qquad (9)$$

## 5.2   Connection to the PH model

The concordance index can be interpreted as the probability of correct ranking (as defined by the given order graph) given a function $f$. Its probabilistic version can thus be cast as a likelihood. Under the assumption that each pair $(j, i)$ is independent of any other pair, the log-likelihood reads

$$\mathcal{L}(f_w, \mathcal{D}, \mathcal{G}) = \log \prod_{\mathcal{E}_{ij}} \Pr\left[f_w(x_i) < f_w(x_j)|w\right]. \qquad (10)$$

As this independence assumption obviously does not hold among all pairs due to transitivity (even though the individual samples $i$ are assumed i.i.d.), it provides a lower bound on the concordance index.

While the probability of correct pairwise ordering, $\Pr\left[f_w(x_i) < f_w(x_j)|w\right]$, is often chosen to be sigmoid in the ranking literature [1], we show in the following that the sigmoid function arises naturally in the context of PH models. Let $T(w^\top x)$ denote the survival time for the patient with covariates $x$ or relative log-hazard $w^\top x$. A larger hazard corresponds to a smaller survival time, cf. Sec. 2. Hence

$$\Pr\left[f_w(x_i) < f_w(x_j)|w\right] = \Pr[T(w^\top x_j) > T(w^\top x_i)|w] = \int_0^\infty \Pr[T(w^\top x_j) > t]p(t|x_i)dt$$

$$= \int_0^\infty S(t|x_j)p(t|x_i)dt = \int_0^\infty -S(t|x_j)S'(t|x_i)dt,$$

where $p(t|x_i)$ is the density function of $T$ for patient $i$ with covariate $x_i$, and $S(t|x_i)$ is the corresponding survival function; $S'(t) = \mathrm{d}S(t)/\mathrm{d}t = -p(t)$. Using Eq. 2 of the PH model, we continue the manipulations:

$$\Pr\left[f_w(x_i) < f_w(x_j)|w\right] = -e^{w^\top x_i} \int_0^\infty e^{-\Lambda_0(t)\left\{e^{w^\top x_j}+e^{w^\top x_i}\right\}} \Lambda_0'(t)dt$$

$$= \frac{e^{w^\top x_i}}{e^{w^\top x_j} + e^{w^\top x_i}} = \sigma[w^\top(x_i - x_j)]. \qquad (11)$$

This derivation shows that the probability of correct pairwise ordering indeed follows the sigmoid function. Assuming a prior $\Pr[w] = \mathcal{N}(w|0, \lambda^{-1})$ for regularization, the optimal *maximum a-posteriori* (MAP) estimator is of the form $\widehat{w}_{\text{MAP}} = \arg\max L(w)$, where the posterior $L(w)$ takes the form of a penalized log-likelihood:

$$L(w) = -\frac{\lambda}{2}\|w\|^2 + \sum_{\mathcal{E}_{ij}} \log \sigma\left[w^T(x_j - x_i)\right]. \qquad (12)$$

This expression is equivalent to (8) except for a few constants that are irrelevant for optimization problem, which justifies our choice of regularization in Eq. 8.

## 6 Exponential lower bound

The exponential $1 - e^{-z}$ can serve as an alternative lower bound on the step indicator function (see Fig. 1(c)). The concordance index can then be lower-bounded by

$$c \geq \frac{1}{|\mathcal{E}|} \sum_{\mathcal{E}_{ij}} 1 - e^{-[f(x_j) - f(x_i)]} \equiv \widehat{c}_{\mathrm{E}}. \tag{13}$$

Analogous to the log-sigmoid bound, for the linear ranking function $f_w(x) = w^\top x$, the lower bound $\widehat{c}_{\mathrm{E}}$ simplifies to

$$\widehat{c}_{\mathrm{E}}(w) = \frac{1}{|\mathcal{E}|} \sum_{\mathcal{E}_{ij}} 1 - e^{-w^\top (x_j - x_i)}, \tag{14}$$

and, penalizing functions with large norm $w$, the regularized version reads

$$\widehat{c}_{\mathrm{Ereg}}(w) = -\frac{\lambda}{2} \|w\|^2 + \frac{1}{|\mathcal{E}|} \sum_{\mathcal{E}_{ij}} 1 - e^{-w^\top (x_j - x_i)}. \tag{15}$$

## 7 Gradient based learning

In order to maximize the regularized concave surrogate we can use any gradient-based learning technique. We use the Polak-Ribière variant of nonlinear *conjugate gradients* (CG) algorithm [11]. The CG method only needs the gradient $g(w)$ and does not require evaluation of the function. It also avoids the need for computing the second derivatives. The convergence of CG is much faster than that of the steepest descent. Using the fact that $d\sigma(z)/dz = \sigma(z)[1 - \sigma(z)]$ and $1 - \sigma(z) = \sigma(-z)$, the gradient of Eq. 9 (log-sigmoid bound) is given by $\nabla_w \widehat{c}_{\mathrm{LSreg}}(w) = -\lambda w - \frac{1}{|\mathcal{E}| \log 2} \sum_{\mathcal{E}_{ij}} (x_i - x_j) \sigma [w^T (x_i - x_j)]$, and the gradient of Eq. 15 (exponential bound) by $\nabla_w \widehat{c}_{\mathrm{Ereg}}(w) = -\lambda w - \frac{1}{|\mathcal{E}|} \sum_{\mathcal{E}_{ij}} (x_i - x_j) e^{-w^\top (x_j - x_i)}$.

## 8 Is Cox's partial likelihood a lower bound on the CI ?

Our experimental results (Sec. 9) indicate that the Coxs method and our proposed methods showed similar performance when assessed using the CI. While our proposed method was formulated to explicitly maximize a lower bound on the concordance index, the Coxs method maximized the partial likelihood. One suspects whether Coxs partial likelihood itself is a lower bound on the concordance index. The argument presented below could give an indication as to why a method which maximizes the partial likelihood also ends up (approximately) maximizing the concordance index. We re-write the exponential bound on the CI for proportional hazard models from Sec. 6

$$\widehat{c}_{\mathrm{E}}(w) = \frac{1}{|\mathcal{E}|} \sum_{T_i \text{ uncensored}} \sum_{T_j \geq T_i} 1 - e^{-w^\top (x_i - x_j)} = 1 - \frac{1}{|\mathcal{E}|} \sum_{T_i \text{ uncensored}} e^{-w^\top x_i} \Big[ \sum_{T_j \geq T_i} e^{w^\top x_j} \Big]$$

$$= 1 - \frac{N_o}{|\mathcal{E}|} \left( \frac{1}{N_o} \sum_{T_i \text{ uncensored}} 1/z_i \right), \quad \text{where} \quad z_i = \frac{e^{w^\top x_i}}{\sum_{T_j \geq T_i} e^{w^\top x_j}} \in [0, 1]. \tag{16}$$

Note that we have replaced $T_j > T_i$ by $T_j \geq T_i$, assuming that there are no ties in the data, *i.e.*, no two survival times are identical, analogous to Cox's partial likelihood approach (cf. Sec. 2.4). The number of uncensored observations is denoted by $N_o$. The Cox's partial likelihood can be written in terms of $z_i$ as $L(w) = \prod_{T_i \text{ uncensored}} z_i = \langle z_i \rangle_{\mathrm{geom}}^{N_o}$, where $\langle z_i \rangle_{\mathrm{geom}}$ denotes the geometric mean of the $z_i$ with uncensored $T_i$. Using the inequality $z_i \geq \min z_i$ the concordance index can be bounded as

$$c \geq 1 - \frac{N_o}{|\mathcal{E}|} \frac{1}{\min z_i}. \tag{17}$$

This says maximizing $\min z_i$ maximizes a lower bound on the concordance index. While this does not say anything about the Cox's partial likelihood it still gives a useful insight. Since $\max z_i = 1$ (because $z_i = 1$ for the largest uncensored $T_i$), maximizing $\min z_i$ can be expected to approximately maximize the geometric mean of $z_i$, and hence the Cox's partial likelihood.

Table 1: *Summary of the five data sets used.* $N$ is the number of patients. $d$ is the number of covariates used.

| Dataset | $N$ | $d$ | Missing | Censored |
|---------|-----|-----|---------|----------|
| MAASTRO | 285 | 19 | 3.6% | 30.5% |
| SUPPORT-1 | 477 | 26 | 14.9% | 36.4% |
| SUPPORT-2 | 314 | 26 | 16.6% | 43.0% |
| SUPPORT-4 | 149 | 26 | 22.0% | 10.7% |
| MELANOMA | 191 | 4 | 0.0% | 70.2% |

## 9  Experiments

In this section we compare the performance of the two different lower bounds on the CI—the log-sigmoid, exponential, and Cox's partial likelihood—on five medical data sets.

### 9.1  Medical datasets

Table 1 summarizes the five data sets we used in our experiments. A substantial amount of data is censored and also missing. The MAASTRO dataset concerns the survival time of non-small cell lung cancer patients, which we analyzed as part of our collaboration. The other medical data sets are publicly available: The SUPPORT dataset [2] is a random sample from Phases I and II of the SUPPORT [9](Study to Understand Prognoses Preferences Outcomes and Risks of Treatment) study. As suggested in [6] we split the dataset into three different datasets, each corresponding to a different cause of death. The MELANOMA data [3] is from a clinical study of skin cancer.

### 9.2  Evaluation procedure

For each data set, $70\%$ of the examples were used for training and the remaining $30\%$ as the hold-out set for testing. We chose the optimal value of regularization parameter $\lambda$ (cf. Eqs. 9 and 15) based on five-fold cross validation on the training set. The tolerance for the conjugate gradient procedure was set to $10^{-3}$. The conjugate-gradient optimization procedure was initialized to the zero vector. All the covariates were normalized to have zero mean and unit variance. As missing values were not the focus of this paper, we used a simple imputation technique. For each missing value, we imputed a sample drawn from a Gaussian distribution with its mean and variance estimated from the available values of the other patients.

### 9.3  Results

The performance was evaluated in terms of the concordance index and the results are tabulated in Table 2. We compare the following methods–(1) Cox's partial likelihood method, and (2) the proposed ranking methods with log-sigmoid and exponential lower bounds. The following observations can be made–(1) The proposed linear ranking method performs slightly better than the Cox's partial likelihood method, but the difference does not appear significant. This agrees with our insights that Cox's partial likelihood may also end up maximizing the CI. (2) The exponential bound shows slightly better performance than the log-sigmoid bound, which may indicate that the tightness of the bound for positive $z$ in Fig. 1(c) is more important than for negative $z$ in our data sets. However the difference is not significant.

## 10  Conclusions

In this paper, we outlined several approaches for maximizing the concordance index, the standard performance measure in survival analysis when cast as a ranking problem. We showed that, for the widely-used proportional hazard models, the log-sigmoid function arises as a natural lower bound on the concordance index. We presented an approach for directly optimizing this lower bound in a computationally efficient way. This optimization procedure can also be applied to other lower bounds, like the exponential one. Apart from that, we showed that maximizing Cox's partial likelihood can be understood as (approximately) maximizing a lower bound on the concordance index, which explains the high CI-scores of proportional hazard models observed in practice. Optimization of each of these three lower bounds results in about the same CI-score in our experiments, with our new approach giving tentatively better results.

Table 2: *Concordance indices for the different methods and datasets.* The mean and the standard deviation are computed over a five fold cross-validation. The results are also shown for a fixed holdout set.

|  | CI for **training set** mean [± std] | CI for **test set** mean [± std] | CI for **holdout set** |
|---|---|---|---|
| **MAASTRO** | | | |
| Cox PH | 0.65 [±0.02] | 0.57 [±0.09] | 0.64 |
| log-sigmoid | 0.69 [±0.02] | 0.60 [±0.06] | 0.64 |
| exponential | 0.69 [±0.02] | 0.64 [±0.08] | 0.65 |
| **SUPPORT-1** | | | |
| Cox PH | 0.76 [±0.01] | 0.74 [±0.05] | 0.79 |
| log-sigmoid | 0.83 [±0.01] | 0.77 [±0.04] | 0.79 |
| exponential | 0.83 [±0.01] | 0.79 [±0.02] | 0.82 |
| **SUPPORT-2** | | | |
| Cox PH | 0.70 [±0.02] | 0.63 [±0.06] | 0.69 |
| log-sigmoid | 0.79 [±0.01] | 0.68 [±0.06] | 0.65 |
| exponential | 0.78 [±0.02] | 0.68 [±0.09] | 0.70 |
| **SUPPORT-4** | | | |
| Cox PH | 0.78 [±0.01] | 0.68 [±0.09] | 0.64 |
| log-sigmoid | 0.80 [±0.01] | 0.74 [±0.12] | 0.71 |
| exponential | 0.79 [±0.01] | 0.73 [±0.03] | 0.71 |
| **MELANOMA** | | | |
| Cox PH | 0.63 [±0.03] | 0.62 [±0.09] | 0.54 |
| log-sigmoid | 0.76 [±0.02] | 0.70 [±0.10] | 0.55 |
| exponential | 0.76 [±0.01] | 0.65 [±0.11] | 0.55 |

## Acknowledgements

We are grateful to R. Bharat Rao for encouragement and support of this work, and to the anonymous reviewers for their valuable comments.

## Footnotes

[1]Generalization to non-linear functions can be achieved easily by using kernels: the linear ranking function class $\mathcal{F}$ is replaced by $\mathcal{H}$, a reproducing kernel Hilbert space (RKHS). The ranking function then is of the form $f(x) = \sum_{i=1}^{N} \alpha_i k(x, x_i)$ where $k$ is the kernel of the RHKS $\mathcal{H}$.

[2]`http://biostat.mc.vanderbilt.edu/twiki/bin/view/Main/DataSets`.

[3]`www.stat.uni-muenchen.de/service/datenarchiv/melanoma/melanoma_e.html`

## References

[1] C.J.C. Burges, T. Shaked, E. Renshaw, A. Lazier, M. Deeds, N. Hamilton, and G. Hullender. Learning to rank using gradient descent. In *Proceeding of the 22th International Conference on Machine Learning*, 2005.

[2] D. R. Cox. Regression models and life-tables (with discussion). *Journal of the Royal Statistical Society, Series B*, 34(2):187–220, 1972.

[3] D. R. Cox. Partial likelihood. *Biometrika*, 62(2):269–276, 1975.

[4] D. R. Cox and D. Oakes. *Analysis of survival data*. Chapman and Hall, 1984.

[5] Y. Freund, R. Iyer, and R. Schapire. An efficient boosting algorithm for combining preferences. *Journal of Machine Learning Research*, 4:933–969, 2003.

[6] F. E. Harrell Jr. *Regression Modeling Strategies, With Applications to Linear Models, Logistic Regression, and Survival Analysis*. Springer, 2001.

[7] R. Herbrich, T. Graepel, P. Bollmann-Sdorra, and K. Obermayer. Learning preference relations for information retrieval. *ICML-98 Workshop: Text Categorization and Machine Learning*, pages 80–84, 1998.

[8] J. D. Kalbfleisch and R. L. Prentice. *The statistical analysis of failure time data*. Wiley-Interscience, 2002.

[9] W.A. Knaus, F. E. Harrell, J. Lynn, et al. The support prognostic model: Objective estimates of survival for seriously ill hospitalized adults. *Annals of Internal Medicine*, 122:191–203, 1995.

[10] H. B. Mann and D. R. Whitney. On a Test of Whether one of Two Random Variables is Stochastically Larger than the Other. *The Annals of Mathematical Statistics*, 18(1):50–60, 1947.

[11] J. Nocedal and S. J. Wright. *Numerical Optimization*. Springer, 1999.

[12] V. C. Raykar, R. Duraiswami, and B. Krishnapuram. A fast algorithm for learning large scale preference relations. In M. Meila and X. Shen, editors, *Proceedings of the Eleventh International Conference on Artificial Intelligence and Statistics*, pages 385–392, 2007.

[13] F. Wilcoxon. Individual comparisons by ranking methods. *Biometrics Bulletin*, 1(6):80–83, December 1945.

